# Sparse probabilistic projections

**Cédric Archambeau**
Department of Computer Science
University College London, United Kingdom
c.archambeau@cs.ucl.ac.uk

**Francis R. Bach**
INRIA - Willow Project
Ecole Normale Supérieure, Paris, France
francis.bach@mines.org

## Abstract

We present a generative model for performing sparse probabilistic projections, which includes sparse principal component analysis and sparse canonical correlation analysis as special cases. Sparsity is enforced by means of automatic relevance determination or by imposing appropriate prior distributions, such as generalised hyperbolic distributions. We derive a variational Expectation-Maximisation algorithm for the estimation of the hyperparameters and show that our novel probabilistic approach compares favourably to existing techniques. We illustrate how the proposed method can be applied in the context of cryptoanalysis as a preprocessing tool for the construction of template attacks.

## 1 Introduction

Principal component analysis (PCA) is widely used for data pre-processing, data compression and dimensionality reduction. However, PCA suffers from the fact that each principal component is a linear combination of all the original variables. It is thus often difficult to interpret the results. In recent years, several methods for sparse PCA have been designed to find projections which retain maximal variance, while enforcing many entries of the projection matrix to be zero [20, 6]. While most of these methods are based on convex or partially convex relaxations of the sparse PCA problem, [16] has looked at using the probabilistic PCA framework of [18] along with $\ell^1$-regularisation. Canonical correlation analysis (CCA) is also commonly used in the context for dimensionality reduction.The goal is here to capture features that are common to several views of the same data. Recent attempts for constructing sparse CCA include [10, 19].

In this paper, we build on the probabilistic interpretation of CCA outlined by [2] and further extended by [13]. We introduce a general probabilistic model, which allows us to infer from an arbitrary number of views of the data, both a shared latent representation and individual low-dimensional representations of each one of them. Hence, the probabilistic reformulations of PCA and CCA fit this probabilistic framework. Moreover, we are interested in sparse solutions, as these are important for interpretation purposes, denoising or feature extraction. We consider a Bayesian approach to the problem. A proper probabilistic approach allows us to treat the trade-off between the modelling accuracy (of the high-dimensional observations by low-dimensional latent variables) and the degree of sparsity of the projection directions in principled way. For example, we do not need to estimate the sparse components successively, using, e.g., deflation, but we can estimate all sparse directions jointly as we are taking the uncertainty of the latent variable into account.

In order to ensure sparse solutions we propose two strategies. The first one, discussed in Appendix A, is based on automatic relevance determination (ARD) [14]. No parameter needs to be set in advance. The entries in the projection matrix which are not well determined by the data are automatically driven to zero. The second approach uses priors from the generalised hyperbolic family [3], and more specifically the inverse Gamma. In this case, the degree of sparsity can be adjusted, eventually leading to very sparse solutions if desired. For both approaches we derive a variational EM algorithm [15].

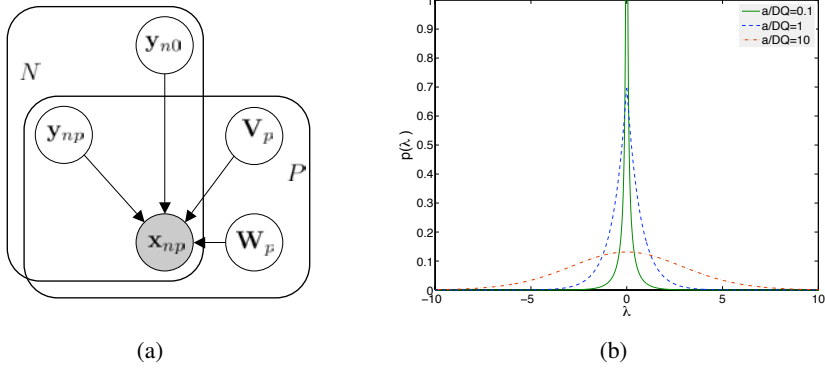

<div align="center">(a)                         (b)</div>

Figure 1: (a) Graphical model (see text for details). Arrows denote conditional dependencies. Shaded and unshaded nodes are respectively observed and unobserved random variables. Plates indicate repetitions. (b) Marginal prior on the individual matrix entries ($b = 1$).

## 2 Generative model

We consider the graphical model shown in Figure 1(a). For each observation, we have $P$ independent measurements $\mathbf{x}_1, \ldots, \mathbf{x}_P$ in different measurement spaces or *views*. The measurement $\mathbf{x}_p \in \mathbb{R}^{D_p}$ is modelled as a mix of a common (or view independent) continuous latent vector $\mathbf{y}_0 \in \mathbb{R}^{Q_0}$ and a view dependent continuous latent vector $\mathbf{y}_p \in \mathbb{R}^{Q_p}$, such that

$$\mathbf{x}_p = \mathbf{W}_p \mathbf{y}_0 + \mathbf{V}_p \mathbf{y}_p + \boldsymbol{\mu}_p + \boldsymbol{\epsilon}_p, \qquad \mathbf{W}_p \in \mathbb{R}^{D_p \times Q_0}, \mathbf{V}_p \in \mathbb{R}^{D_p \times Q_p}, \qquad (1)$$

where $\{\boldsymbol{\mu}_p\}_{p=1}^P$ are the view dependent offsets and $\boldsymbol{\epsilon}_p \sim \mathcal{N}(\mathbf{0}, \tau_p^{-1}\mathbf{I}_{D_p})$ is the residual error in view $p$.

We are interested in the case where $\mathbf{y}_0$ and $\mathbf{y}_p$ are low-dimensional vectors, i.e., $Q_0, Q_p \ll D_p$ for all $p$. We impose Gaussian priors on the latent vectors:

$$\mathbf{y}_0 \sim \mathcal{N}(\mathbf{0}, \boldsymbol{\Phi}_0^{-1}), \qquad \mathbf{y}_p \sim \mathcal{N}(\mathbf{0}, \boldsymbol{\Phi}_p^{-1}), \ p \in \{1, \ldots, P\}. \qquad (2)$$

The resulting generative model comprises a number of popular probabilistic projection techniques as special cases. If there is a single view (and a single latent cause) and the prior covariance is diagonal, we recover probabilistic factor analysis [9]. If the prior is also isotropic, then we get probabilistic PCA [18]. If there are two views, we recover probabilistic CCA [2].

We seek a solution for which the matrices $\{\mathbf{W}_p\}_{p=1}^P$ and $\{\mathbf{V}_p\}_{p=1}^P$ are sparse, i.e. most of their entries are zero. One way to achieve sparsity is by means of ARD-type priors [14]. In this framework, a zero-mean Gaussian prior is imposed on the entries of the weight matrices:

$$w_{i_p j} \sim \mathcal{N}(0, 1/\alpha_{i_p j}), \qquad i_p \in \{1, \ldots, D_p\}, j \in \{1, \ldots, Q_0\}, \qquad (3)$$

$$v_{i_p k_p} \sim \mathcal{N}(0, 1/\beta_{i_p k_p}), \qquad i_p \in \{1, \ldots, D_p\}, k_p \in \{1, \ldots, Q_p\}. \qquad (4)$$

Type II maximum likelihood leads then to a sparse solution when considering independent hyperparameters. The updates arising in the context of probabilistic projections are given in Appendix A. Since marginalisation with respect to both the latent vectors and the weights is intractable, we apply variational EM [15]. Unfortunately, following this route does not allow us to adjust the degree of sparsity, which is important e.g. for interpretation purposes or for feature extraction.

Hence, we seek a more flexible approach. In the remaining of this paper, we will assume that the marginal prior on each weight $\lambda_{ij}$, which is either an entry of $\{\mathbf{W}_p\}_{p=1}^P$ or $\{\mathbf{V}_p\}_{p=1}^P$ and will be defined shortly, has the form of an (infinite) weighted sum of scaled Gaussians:

$$p(\lambda_{ij}) = \int \mathcal{N}(0, \gamma_{ij}^{-1}) \, p(\gamma_{ij}) \, d\gamma_{ij}. \qquad (5)$$

We will chose the prior over $\gamma_{ij}$ in such a way that the resulting marginal prior over the corresponding $\lambda_{ij}$ induces sparsity. A similar approach was followed in the context of sparse nonparametric Bayesian regression in [4, 5].

## 2.1 Compact reformulation of the generative model

Before discussing the approximate inference scheme, we rewrite the model in a more compact way. Let us denote the $n$th observation, the corresponding latent vector and the means respectively by

$$\mathbf{x}_n = \left(\mathbf{x}_{n1}^\top, \ldots, \mathbf{x}_{nP}^\top\right)^\top, \qquad \mathbf{z}_n = \left(\mathbf{y}_{n0}^\top, \mathbf{y}_{n1}^\top, \ldots, \mathbf{y}_{nP}^\top\right)^\top, \qquad \boldsymbol{\mu} = \left(\boldsymbol{\mu}_1^\top, \ldots, \boldsymbol{\mu}_P^\top\right)^\top.$$

The generative model can be reformulated as follows:

$$\mathbf{z}_n \sim \mathcal{N}(\mathbf{0}, \boldsymbol{\Phi}^{-1}), \qquad\qquad \boldsymbol{\Phi} \in \mathbb{R}^{Q \times Q},\ Q = Q_0 + \textstyle\sum_p Q_p, \qquad (6)$$

$$\lambda_{ij}|\gamma_{ij} \sim \mathcal{N}(0, \gamma_{ij}^{-1}), \qquad\qquad i \in \{1, \ldots, D\},\ j \in \{1, \ldots, Q\},\ D = \textstyle\sum_p D_p, \quad (7)$$

$$\mathbf{x}_n|\mathbf{z}_n, \boldsymbol{\Lambda} \sim \mathcal{N}(\boldsymbol{\Lambda}\mathbf{z}_n + \boldsymbol{\mu}, \boldsymbol{\Psi}^{-1}), \qquad \boldsymbol{\Lambda} \in \mathbb{R}^{D \times Q},\ \boldsymbol{\Psi} \in \mathbb{R}^{D \times D}, \qquad (8)$$

where

$$\boldsymbol{\Lambda} = \begin{pmatrix} \boldsymbol{\Lambda}_1 \\ \vdots \\ \boldsymbol{\Lambda}_P \end{pmatrix} = \begin{pmatrix} \mathbf{W}_1 & \mathbf{V}_1 & \ldots & \mathbf{0} \\ \vdots & \vdots & \ddots & \vdots \\ \mathbf{W}_P & \mathbf{0} & \ldots & \mathbf{V}_P \end{pmatrix}, \qquad \boldsymbol{\Psi} = \begin{pmatrix} \tau_1 \mathbf{I}_{D_1} & \ldots & \mathbf{0} \\ \vdots & \ddots & \vdots \\ \mathbf{0} & \ldots & \tau_P \mathbf{I}_{D_P} \end{pmatrix}.$$

Note that we do not assume that the latent spaces are correlated as $\boldsymbol{\Phi} = \mathrm{diag}\{\boldsymbol{\Phi}_0, \boldsymbol{\Phi}_1, \ldots, \boldsymbol{\Phi}_P\}$. This is consistent with the fact the common latent space is modelled independently through $\mathbf{y}_0$. Subsequently, we will also denote the matrix of the hyperparameters by $\boldsymbol{\Gamma} \in \mathbb{R}^{D \times Q}$, where we set (and fix) $\gamma_{ij} = \infty$ for all $\lambda_{ij} = 0$.

## 2.2 Sparsity inducing prior over the individual scale variables

We impose an inverse Gamma prior on the scale variable $\gamma_{ij}$:

$$\gamma_{ij} \sim \mathcal{IG}(a/DQ, b), \qquad (9)$$

for all $i$ and $j$. The shape parameter $a$ and the scale parameter $b$ are non-negative. The marginal prior on the weight $\lambda_{ij}$ is then in the class of the generalised hyperbolic distributions [3] and is defined in terms of the modified Bessel function of the third kind $K_\omega(\cdot)$:

$$p(\lambda_{ij}) = \sqrt{\frac{2}{\pi}} \frac{b^{\frac{a}{DQ}}}{\Gamma(\frac{a}{DQ})} \left(\frac{\lambda_{ij}^2}{2b}\right)^{\frac{a}{2DQ} - \frac{1}{4}} K_{\frac{a}{DQ} - \frac{1}{2}}\left(\sqrt{2b\lambda_{ij}^2}\right) \qquad (10)$$

for $\lambda_{ij} \neq 0$, and

$$\lim_{\lambda_{ij} \to 0} p(\lambda_{ij}) = \begin{cases} \sqrt{\frac{b}{2\pi}} \frac{\Gamma(\frac{a}{DQ} - \frac{1}{2})}{\Gamma(\frac{a}{DQ})} & \frac{a}{DQ} > \frac{1}{2}, \\ \infty & \text{otherwise.} \end{cases} \qquad (11)$$

The function $\Gamma(\cdot)$ is the (complete) Gamma function.

The effective prior on the individual weights is shown in Figure 1(b). Intuitively, the joint distribution over the weights is sparsity inducing as it is sharply peaked around zero (and in fact infinite for sufficiently small $a$). It favours only a small number of weights to be non-zero if the scale variable $b$ is sufficiently large. For a more formal discussion in the context of regression we refer to [7].

It is interesting to note that for $a/DQ = 1$ we recover the popular Laplace prior, which is equivalent to the $\ell^1$-regulariser or the LASSO [17], and for $a/DQ \to 0$ and $b \to 0$ the resulting prior is the Normal-Jeffreys prior. In fact, the automatic thresholding method described in Appendix A fits also into the framework defined by (5). However, it corresponds to imposing a flat prior on the scale variables over the log-scale, which is a limiting case of the Gamma distribution. When imposing independent Gamma priors on the scale variables, the effective joint marginal is a product of Student-$t$ distributions, which again is sharply peaked around zero and sparsity inducing.

## 3 Variational approximation

We view $\{\mathbf{z}_n\}_{n=1}^N$ and matrix $\boldsymbol{\Gamma}$ as latent variables, and optimise the parameters $\boldsymbol{\theta} = \{\boldsymbol{\mu}, \boldsymbol{\Phi}, \boldsymbol{\Lambda}, \boldsymbol{\Psi}\}$ by EM. In other words, we view the weight matrix $\boldsymbol{\Lambda}$ as a matrix of parameter and estimate the

entries by maximum a posteriori (MAP) learning. The other parameters are estimated by maximum likelihood (ML).

The variational free energy is given by

$$\mathcal{F}_q(\mathbf{x}_1, \ldots, \mathbf{x}_N, \boldsymbol{\theta}) = -\sum_{n=1}^{N} \langle \ln p(\mathbf{x}_n, \mathbf{z}_n, \boldsymbol{\Gamma}|\boldsymbol{\theta}) \rangle_q - \mathrm{H}[q(\mathbf{z}_1, \ldots, \mathbf{z}_N, \boldsymbol{\Gamma})], \qquad (12)$$

where $\langle \cdot \rangle_q$ denotes the expectation with respect to the variational distribution $q$ and $\mathrm{H}[\cdot]$ is the differential entropy. Since the Kullback-Leibler divergence (KL) is non-negative, the negative free energy is a lower bound to log-marginal likelihood:

$$\sum_{n=1}^{N} \ln p(\mathbf{x}_n|\boldsymbol{\theta}) = -\mathcal{F}_q(\{\mathbf{x}_n\}, \boldsymbol{\theta}) + \mathrm{KL}\left[q(\{\mathbf{z}_n\}, \boldsymbol{\Gamma}) \| p(\{\mathbf{z}_n\}, \boldsymbol{\Gamma})|\{\mathbf{x}_n\}, \boldsymbol{\theta})\right] \geqslant -\mathcal{F}_q(\{\mathbf{x}_n\}, \boldsymbol{\theta}). \qquad (13)$$

Interestingly it is not required to make a factorised approximation of the the joint posterior $q$ to find a tractable solution. Indeed, the posterior $q$ factorises naturally given the data and the weights, such that the posteriors we will obtain in the E-step are exact.

The variational EM finds maximum likelihood estimates for the parameters by cycling through the following two steps until convergence:

1. The posterior over the latent variables are computed for fixed parameters by minimising the KL in (13). It can be shown that the variational posteriors are given by

$$q(\mathbf{z}_1, \ldots, \mathbf{z}_N) \propto \prod_{n=1}^{N} e^{\langle \ln p(\mathbf{x}_n, \mathbf{z}_n, \boldsymbol{\Gamma}|\boldsymbol{\theta}) \rangle_{q(\boldsymbol{\Gamma})}}, \qquad (14)$$

$$q(\boldsymbol{\Gamma}) \propto e^{\langle \ln p(\mathbf{x}_n, \mathbf{z}_n|\boldsymbol{\Gamma}, \boldsymbol{\theta}) \rangle_{q(\mathbf{z}_1, \ldots, \mathbf{z}_N)}} p(\boldsymbol{\Gamma}). \qquad (15)$$

2. The variational free energy (12) is minimised wrt the parameters for fixed $q$. This leads in effect to type II ML estimates for the paramteres and is equivalent to maximising the expected complete log-likelihood:

$$\boldsymbol{\theta} \leftarrow \underset{\boldsymbol{\theta}}{\mathrm{argmax}} \sum_{n=1}^{N} \langle \ln p(\mathbf{x}_n, \mathbf{z}_n, \boldsymbol{\Gamma}|\boldsymbol{\theta}) \rangle_q. \qquad (16)$$

Depending on the initialisation, the variational EM algorithm converges to a local maximum of the log-marginal likelihood. The convergence can be checked by monitoring the variational lower bound, which monotonically increases during the optimisation. The explicit expression of the variational bound is here omitted due to a lack of space

## 3.1 Posterior of the latent vectors

The joint posterior of the latent vectors factorises into $N$ posteriors due to the fact the observations are independent. Hence, the posterior of each low-dimenstional latent vector is given by

$$q(\mathbf{z}_n) = \mathcal{N}(\bar{\mathbf{z}}_n, \bar{\mathbf{S}}_n), \qquad (17)$$

where $\bar{\mathbf{z}}_n = \bar{\mathbf{S}}_n \boldsymbol{\Lambda}^\top \boldsymbol{\Psi}(\mathbf{x}_n - \boldsymbol{\mu})$ is the mean and $\bar{\mathbf{S}}_n = \left(\boldsymbol{\Lambda}^\top \boldsymbol{\Psi}\boldsymbol{\Lambda} + \boldsymbol{\Phi}\right)^{-1}$ is the covariance.

## 3.2 Posterior of the scale variables

The inverse Gamma distribution is not conjugate to the exponential family. However, the posterior over matrix $\boldsymbol{\Gamma}$ is tractable. It has the form of a product of generalised inverse Gaussian distributions (see Appendix B for a formal definition):

$$q(\boldsymbol{\Gamma}) = \prod_{i=1}^{D} \prod_{j=1}^{Q} p(\gamma_{ij}|\lambda_{ij}) = \prod_{i=1}^{D} \prod_{j=1}^{Q} \mathcal{N}^{-}(\bar{\omega}_{ij}, \bar{\varphi}_{ij}, \bar{\chi}_{ij}) \qquad (18)$$

where $\bar{\omega}_{ij} = -\frac{a}{DQ} + \frac{1}{2}$ is the index and $\bar{\varphi}_{ij} = \lambda_{ij}^2$ and $\bar{\chi}_{ij} = 2b$ are the shape parameters. The factorised form arises from the scale variable being independent conditioned on the weights.

### 3.3 Update for the parameters

Based on the properties of the Gaussian and the generalised inverse Gaussian, we can compute the variational lower bound, which can then be maximised. This leads to the following updates:

$$\boldsymbol{\mu} \leftarrow \frac{1}{N} \sum_{n=1}^{N} (\mathbf{x}_n - \boldsymbol{\Lambda}\bar{\mathbf{z}}_n), \qquad \boldsymbol{\Phi}^{-1} \leftarrow \frac{1}{N} \sum_{n=1}^{N} \text{diag}\{\bar{\mathbf{z}}_n \bar{\mathbf{z}}_n^\top + \bar{\mathbf{S}}_n\}, \tag{19}$$

$$\boldsymbol{\lambda}_i \leftarrow \left(\bar{\boldsymbol{\Gamma}}_i + \boldsymbol{\Psi}(i,i) \sum_{n=1}^{N} \langle \mathbf{z}_n \mathbf{z}_n^\top \rangle\right)^{-1} \boldsymbol{\Psi}(i,i) \sum_{n=1}^{N} (\mathbf{x}_n(i) - \boldsymbol{\mu}(i))\bar{\mathbf{z}}_n, \tag{20}$$

$$\tau_p^{-1} \leftarrow \frac{1}{ND_p} \sum_{n=1}^{N} \left\{ (\mathbf{x}_{np} - \boldsymbol{\mu}_p)^\top (\mathbf{x}_{np} - \boldsymbol{\mu}_p) - 2(\mathbf{x}_{np} - \boldsymbol{\mu}_p)^\top \boldsymbol{\Lambda}_p \bar{\mathbf{z}}_n + \langle (\boldsymbol{\Lambda}_p \mathbf{z}_n)^\top \boldsymbol{\Lambda}_p \mathbf{z}_n \rangle \right\}, \tag{21}$$

where the required expectations are given by

$$\langle \mathbf{z}_n \mathbf{z}_n^\top \rangle = \bar{\mathbf{S}}_n + \bar{\mathbf{z}}_n \bar{\mathbf{z}}_n^\top, \qquad\qquad \langle (\boldsymbol{\Lambda}_p \mathbf{z}_n)^\top \boldsymbol{\Lambda}_p \mathbf{z}_n \rangle = \text{tr}\{\langle \bar{\mathbf{z}}_n \bar{\mathbf{z}}_n^\top \rangle \boldsymbol{\Lambda}_p^\top \boldsymbol{\Lambda}_p\}, \tag{22}$$

$$\bar{\boldsymbol{\Gamma}}_i = \text{diag}\{\langle \gamma_{i1} \rangle, \ldots, \langle \gamma_{iQ} \rangle\}, \qquad\qquad \langle \gamma_{ij} \rangle = \sqrt{\frac{\bar{\chi}_{ij}}{\bar{\varphi}_{ij}}} \frac{K_{\omega+1}\left(\sqrt{\bar{\chi}_{ij}\bar{\varphi}_{ij}}\right)}{K_\omega\left(\sqrt{\bar{\chi}_{ij}\bar{\varphi}_{ij}}\right)}. \tag{23}$$

Note that $\text{diag}\{\cdot\}$ denotes a block-diagonal operation in (19). More importantly, since we are seeking a sparse projection matrix, we do not suffer from the rotational ambiguity problem as is for example the case standard probabilistic PCA.

## 4 Experiments

### 4.1 Synthetic denoising experiments

Because of identifiability issues which are subject of ongoing work, we prefer to compare various methods for sparse PCA in a denoising experiment. That is, we assume that the data were generated from sparse components plus some noise and we compare the various sparse PCA on the denoising task, i.e., on the task of recovering the original data. We generated the data as follows: select uniformly at random $M = 4$ unit norm sparse vectors in $P = 10$ dimensions with known number $S = 4$ of non zero entries, then generate i.i.d. values of the random variables $Z$ from three possible distributions (Gaussian, Laplacian, uniform), then add isotropic noise of relative standard deviation $1/2$. When the latent variables are Gaussian, our model exactly matches the data and our method should provide a better fit; however, we consider also situations where the model is misspecified in order to study the robustness of our probabilistic model.

We consider our two models: SCA-1 (which uses automatic relevance determination type of sparsity priors) and SCA-2 (which uses generalised hyperbolic distribution), where we used 6 latent dimensions (larger than 4) and fixed hyperparameters that lead to vague priors. Those two models thus have no free parameters and we compare them to the following methods, which all have two regularisation parameters (rank and regularisation): DSPCA [6], the method of Zou [20] and the recent method of [16] which essentially considers a probabilistic PCA with $\ell^1$-penalty on the weights.

In Table 1 we report mean-squared reconstruction error averaged over 10 replications. It can be seen that two proposed probabilistic approaches perform similarly and significantly outperform other sparse PCA methods, even when the model is misspecified.

### 4.2 Template attacks

Power consumption and electromagnetic radiation are among the most extensively used side-channels for analysing physically observable cryptographic devices. A common belief is that the useful information for attacking a device is hidden at times where the traces (or time series) have large variance. Once the relevant samples have been identified they can be used to construct templates, which can then be used to assess if a device is secure. A simple, yet very powerful approach, recently proposed by [1], is to select time samples based on PCA. Figure 2(a) shows the weight

| $N$ | SCA-1 | SCA-2 | Zou | DSPCA | L1-PCA |
|---|---|---|---|---|---|
| 100 | 39.9 | 40.8 | 42.2 | 42.9 | 50.8 |
| 200 | 36.5 | 36.8 | 40.8 | 41.4 | 50.4 |
| 400 | 35.5 | 35.5 | 39.8 | 40.3 | 42.5 |
| $N$ | SCA-1 | SCA-2 | Zou | DSPCA | L1-PCA |
| 100 | 39.9 | 40.9 | 42.6 | 43.6 | 49.8 |
| 200 | 36.8 | 37.0 | 40.9 | 41.1 | 48.1 |
| 400 | 36.4 | 36.4 | 40.5 | 40.7 | 46.8 |
| $N$ | SCA-1 | SCA-2 | Zou | DSPCA | L1-PCA |
| 100 | 39.3 | 40.3 | 42.7 | 43.4 | 48.5 |
| 200 | 36.5 | 36.7 | 40.2 | 40.8 | 46.2 |
| 300 | 35.8 | 35.8 | 40.6 | 40.9 | 41.0 |

Table 1: Denoising experiment with sparse PCA (we report mean squared errors): (top) Gaussian distributed latent vectors, (middle) latent vectors generated from the uniform distribution, (bottom) latent vectors generated from the Laplace distribution.

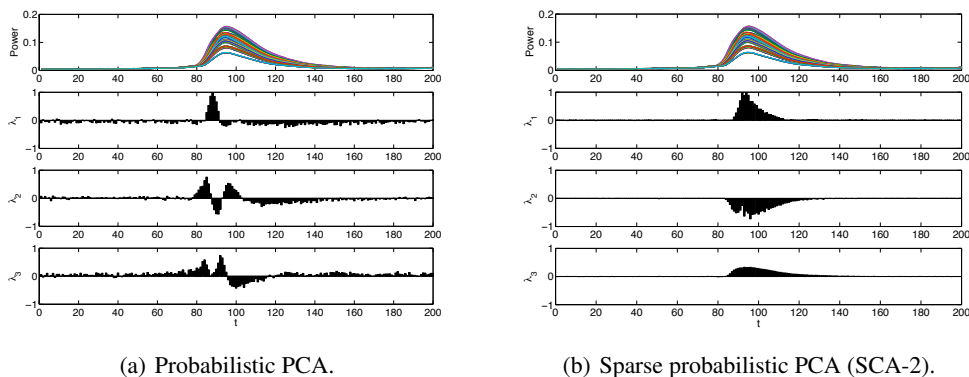

(a) Probabilistic PCA.     (b) Sparse probabilistic PCA (SCA-2).

Figure 2: Power traces and first three principal directions.

associated to each time sample by the first three principal directions found by PCA. The problem with this approach is that all time samples get a non-zero weights. As a result, the user has to define a threshold manually in order to decide whether the information leakage at time $t$ is relevant or not. Figure 2(b) shows the weight associated to the time samples by SCA-2 when using a Laplace prior (i.e. for $a/DQ = 1$). It can be observed that one gets a much better picture of where the relevant information is. Clearly, sparse probabilitic PCA can be viewed as being more robust to spurious noise and provides a more reliable and amenable solution.

## 5 Conclusion

In this paper we introduced a general probabilistic model for inferring sparse probabilistic projection matrices. Sparsity was enforced by either imposing an ARD-type prior or by means of the a Normal-Inverse Gamma prior. Although the inverse Gamma is not conjugate to the exponential family, the posterior is tractable as it is a special case of the generalised inverse Gaussian [12], which in turn is a conjugate prior to this family. Future work will include the validation of the method on a wide range of applications and in particular as a feature extracting tool.

**Acknowledgments**

We are grateful to the PASCAL European network of excellence for partially supporting this work.

# A  Automatic thresholding the weights by ARD

In this section, we provide the updates for achieving automatic thresholding of projection matrix entries in a probabilistic setting. We apply Tipping's sparse Bayesian theory [8], which is closely related to ARD [14]. More specifically, we assume the prior over the scale variables is uniform over a log-scale, which is a limiting case of the Gamma distribution.

Let us view $\{\mathbf{z}_n\}_{n=1}^N$ and $\boldsymbol{\Lambda}$ as latent variables and optimise the parameters $\boldsymbol{\theta} = \{\boldsymbol{\mu}, \boldsymbol{\Phi}, \boldsymbol{\Psi}, \boldsymbol{\Gamma}\}$ by variational EM. The variational free energy is given by

$$\mathcal{F}_q(\mathbf{x}_1, \ldots, \mathbf{x}_N, \boldsymbol{\theta}) = -\sum_{n=1}^N \langle \ln p(\mathbf{x}_n, \mathbf{z}_n, \boldsymbol{\Lambda} | \boldsymbol{\theta}) \rangle_q - \mathrm{H}[q(\mathbf{z}_1, \ldots, \mathbf{z}_N, \boldsymbol{\Lambda})]. \tag{24}$$

In order to find a tractable solution, we further have to assume that the approximate posterior $q$ has a factorised form. We can then compute the posterior of the low-dimensional latent vectors:

$$q(\mathbf{z}_n) = \mathcal{N}(\bar{\mathbf{z}}_n, \bar{\mathbf{S}}_n), \tag{25}$$

where $\bar{\mathbf{z}}_n = \bar{\mathbf{S}}_n \bar{\boldsymbol{\Lambda}}^\top \boldsymbol{\Psi} (\mathbf{x}_n - \boldsymbol{\mu})$ and $\bar{\mathbf{S}}_n = (\bar{\boldsymbol{\Lambda}}^\top \boldsymbol{\Psi} \bar{\boldsymbol{\Lambda}} + \sum_i \boldsymbol{\Psi}(i,i) \bar{\boldsymbol{\Sigma}}_i + \boldsymbol{\Phi})^{-1}$. And the posterior of the weights is given by

$$q(\boldsymbol{\Lambda}) = \prod_{i=1}^D q(\boldsymbol{\lambda}_i) = \prod_{i=1}^D \mathcal{N}(\bar{\boldsymbol{\lambda}}_i, \bar{\boldsymbol{\Sigma}}_i), \tag{26}$$

where $\bar{\boldsymbol{\lambda}}_i = \bar{\boldsymbol{\Sigma}}_i \boldsymbol{\Psi}(i,i) \sum_n (\mathbf{x}_n(i) - \boldsymbol{\mu}(i)) \bar{\mathbf{z}}_n$ and $\bar{\boldsymbol{\Sigma}}_i = (\boldsymbol{\Gamma}_i + \boldsymbol{\Psi}(i,i) \sum_n \{\bar{\mathbf{S}}_n + \bar{\mathbf{z}}_n \bar{\mathbf{z}}_n^\top\})^{-1}$. The partially factorised form $\prod_i q(\boldsymbol{\lambda}_i)$ arises naturally. Note also that the update for the mean weights has the same form as in (20). Finally, the updates for the parameters are found by maximising the negative free energy, which corresponds to performing type II ML for the scaling variables. This yields

$$\boldsymbol{\mu} \leftarrow \frac{1}{N} \sum_{n=1}^N (\mathbf{x}_n - \bar{\boldsymbol{\Lambda}} \bar{\mathbf{z}}_n), \quad \boldsymbol{\Phi}^{-1} \leftarrow \frac{1}{N} \sum_{n=1}^N \mathrm{diag}\{\bar{\mathbf{z}}_n \bar{\mathbf{z}}_n^\top + \bar{\mathbf{S}}_n\}, \quad \gamma_{ij} \leftarrow \langle \lambda_{ij}^2 \rangle^{-1}, \tag{27}$$

$$\tau_p^{-1} \leftarrow \frac{1}{N D_p} \sum_{n=1}^N \left\{ (\mathbf{x}_{np} - \boldsymbol{\mu}_p)^\top (\mathbf{x}_{np} - \boldsymbol{\mu}_p) - 2(\mathbf{x}_{np} - \boldsymbol{\mu}_p)^\top \bar{\boldsymbol{\Lambda}}_p \bar{\mathbf{z}}_n + \langle (\boldsymbol{\Lambda}_p \mathbf{z}_n)^\top \boldsymbol{\Lambda}_p \mathbf{z}_n \rangle \right\}, \tag{28}$$

where $\langle \lambda_{ij}^2 \rangle = \bar{\lambda}_{ij}^2 + \bar{\boldsymbol{\Sigma}}_i(j,j)$ and $\langle (\boldsymbol{\Lambda}_p \mathbf{z}_n)^\top \boldsymbol{\Lambda}_p \mathbf{z}_n \rangle = \mathrm{tr}\{(\bar{\mathbf{z}}_n \bar{\mathbf{z}}_n^\top + \bar{\mathbf{S}}_n) \sum_{i_p} (\bar{\boldsymbol{\lambda}}_{i_p} \bar{\boldsymbol{\lambda}}_{i_p}^\top + \bar{\boldsymbol{\Sigma}}_{i_p})\}$.

# B  Generalised inverse Gaussian distribution

The Generalised inverse Gaussian distribution is in the class of generalised hyperbolic distributions. It is defined as follows [12, 11]:

$$y \sim \mathcal{N}^-(\omega, \chi, \phi) = \frac{\chi^{-\omega}(\sqrt{\chi\phi})^\omega}{2K_\omega(\sqrt{\chi\phi})} y^{\omega-1} e^{-\frac{1}{2}(\chi y^{-1} + \phi y)}, \tag{29}$$

where $y > 0$ and $K_\omega(\cdot)$ is the modified Bessel function of the third kind[1] with index $\omega$.

The following expectations are useful [12]:

$$\langle y \rangle = \sqrt{\frac{\chi}{\phi}} R_\omega(\sqrt{\chi\phi}), \quad \langle y^{-1} \rangle = \sqrt{\frac{\phi}{\chi}} R_{-\omega}(\sqrt{\chi\phi}), \quad \langle \ln y \rangle = \ln \omega + \frac{d \ln K_\omega(\sqrt{\chi\phi})}{d\omega}, \tag{30}$$

where $R_\omega(\cdot) \equiv K_{\omega+1}(\cdot) / K_\omega(\cdot)$.

**Inverse Gamma distribution**

When $\phi = 0$ and $\omega < 0$, the generalised inverse Gaussian distribution reduces to the inverse Gamma distribution:

$$\mathcal{IG}(a,b) = \frac{b^a}{\Gamma(a)} x^{-a-1} e^{-\frac{b}{x}}, \quad a, b > 0. \tag{31}$$

It is straightforward to verify this result by posing $a = -\omega$ and $b = \chi/2$, and noting that

$$\lim_{y \to 0} K_\omega(y) = \Gamma(-\omega) 2^{-\omega-1} y^\omega \tag{32}$$

for $\omega < 0$.

## Footnotes

[1]The modified Bessel function of the third kind is known under various names. In particular, it is also known as the modified Bessel function of the second kind (cf. E. W. Weisstein: "Modified Bessel Function of the Second Kind." From MathWorld: http://mathworld.wolfram.com/ModifiedBesselFunctionoftheSecondKind.html).

# References

[1] C. Archambeau, E. Peeters, F.-X. Standaert, and J.-J. Quisquater. Template attacks in principal subspaces. In L. Goubin and M. Matsui, editors, *8th International Workshop on Cryptographic Hardware and Embedded Systems(CHES)*, volume 4249 of *Lecture Notes in Computer Science*, pages 1–14. Springer, 2006.

[2] F. Bach and M. I. Jordan. A probabilistic interpretation of canonical correlation analysis. Technical Report 688, Department of Statistics, University of California, Berkeley, 2005.

[3] O. Barndorff-Nielsen and R. Stelzer. Absolute moments of generalized hyperbolic distributions and approximate scaling of normal inverse Gaussian Lévy processes. *Scandinavian Journal of Statistics*, 32(4):617–637, 2005.

[4] P. J. Brown and J. E. Griffin. Bayesian adaptive lassos with non-convex penalization. Technical Report CRiSM 07-02, Department of Statistics, University of Warwick, 2007.

[5] F. Caron and A. Doucet. Sparse bayesian nonparametric regression. In *25th International Conference on Machine Learning (ICML)*. ACM, 2008.

[6] A. d'Aspremont, E. L. Ghaoui, M. I. Jordan, and G. R. G. Lanckriet. A direct formulation for sparse PCA using semidefinite programming. *SIAM Review*, 49(3):434–48, 2007.

[7] J. Fan and R. Li. Variable selection via nonconcave penalized likelihood and its oracle properties. *Journal of the American Statistical Association*, 96:1348–1360, 2001.

[8] A. C. Faul and M. E. Tipping. Analysis of sparse Bayesian learning. In T. G. Dietterich, S. Becker, and Z. Ghahramani, editors, *Advances in Neural Information Processing Systems 14 (NIPS)*, pages 383–389. The MIT Press, 2002.

[9] Z. Ghahramani and G. E. Hinton. The EM algorithm for mixtures of factor analyzers. Technical Report CRG-TR-96-1, Department of Computer Science, University of Toronto, 1996.

[10] D. Hardoon and J. Shawe-Taylor. Sparse canonical correlation analysis. Technical report, PASCAL EPrints, 2007.

[11] Wenbo Hu. *Calibration of multivariate generalized hyperbolic distributions using the EM algorithm, with applications in risk management, portfolio optimization and portfolio credit risk*. PhD thesis, Florida State University, United States of America, 2005.

[12] B. Jørgensen. *Statistical Properties of the Generalized Inverse Gaussian Distribution*. Springer-Verlag, 1982.

[13] A. Klami and S. Kaski. Local dependent components. In Z. Ghahramani, editor, *24th International Conference on Machine Learning (ICML)*, pages 425–432. Omnipress, 2007.

[14] D. J. C. MacKay. Bayesian methods for backprop networks. In E. Domany, J.L. van Hemmen, and K. Schulten, editors, *Models of Neural Networks, III*, pages 211–254. 1994.

[15] R. M. Neal and G. E. Hinton. A view of the EM algorithm that justifies incremental, sparse, and other variants. In M. I. Jordan, editor, *Learning in Graphical Models*, pages 355–368. The MIT press, 1998.

[16] C. D. Sigg and J. M. Buhmann. Expectation-maximization for sparse and non-negative PCA. In *25th International Conference on Machine Learning (ICML)*. ACM, 2008.

[17] R. Tibshirani. Regression shrinkage and selection via the LASSO. *Journal of the Royal Statistical Society B*, 58:267–288, 1996.

[18] M. E. Tipping and C. M. Bishop. Probabilistic principal component analysis. *Journal of the Royal Statistical Society B*, 61:611–622, 1999.

[19] D. Torres, D. Turnbull, B. K. Sriperumbudur, L. Barrington, and G.Lanckriet. Finding musically meaningful words using sparse CCA. In *NIPS workshop on Music, Brain and Cognition*, 2007.

[20] H. Zou, T. Hastie, and R. Tibshirani. Sparse principal component analysis. *Journal of Computational and Graphical Statistics*, 15(2):265–286, 2006.

